# Active learning of neural response functions with Gaussian processes

**Mijung Park**
Electrical and Computer Engineering
The University of Texas at Austin
mjpark@mail.utexas.edu

**Greg Horwitz**
Departments of Physiology and Biophysics
The University of Washington
ghorwitz@uw.edu

**Jonathan W. Pillow**
Departments of Psychology and Neurobiology
The University of Texas at Austin
pillow@mail.utexas.edu

## Abstract

A sizeable literature has focused on the problem of estimating a low-dimensional feature space for a neuron's stimulus sensitivity. However, comparatively little work has addressed the problem of estimating the nonlinear function from feature space to spike rate. Here, we use a Gaussian process (GP) prior over the infinite-dimensional space of nonlinear functions to obtain Bayesian estimates of the "nonlinearity" in the linear-nonlinear-Poisson (LNP) encoding model. This approach offers increased flexibility, robustness, and computational tractability compared to traditional methods (e.g., parametric forms, histograms, cubic splines). We then develop a framework for optimal experimental design under the GP-Poisson model using *uncertainty sampling*. This involves adaptively selecting stimuli according to an information-theoretic criterion, with the goal of characterizing the nonlinearity with as little experimental data as possible. Our framework relies on a method for rapidly updating hyperparameters under a Gaussian approximation to the posterior. We apply these methods to neural data from a color-tuned simple cell in macaque V1, characterizing its nonlinear response function in the 3D space of cone contrasts. We find that it combines cone inputs in a highly nonlinear manner. With simulated experiments, we show that optimal design substantially reduces the amount of data required to estimate these nonlinear combination rules.

## 1   Introduction

One of the central problems in systems neuroscience is to understand how neural spike responses convey information about environmental stimuli, which is often called the *neural coding problem*. One approach to this problem is to build an explicit encoding model of the stimulus-conditional response distribution $p(r|\mathbf{x})$, where $r$ is a (scalar) spike count elicited in response to a (vector) stimulus $\mathbf{x}$. The popular linear-nonlinear-Poisson (LNP) model characterizes this encoding relationship in terms of a cascade of stages: (1) linear dimensionality reduction using a bank of filters or *receptive fields*; (2) a nonlinear function from filter outputs to spike rate; and (3) an inhomogeneous Poisson spiking process [1].

While a sizable literature [2–10] has addressed the problem of estimating the linear front end to this model, the nonlinear stage has received comparatively less attention. Most prior work has focused on: simple parametric forms [6, 9, 11]; non-parametric methods that do not scale easily to high

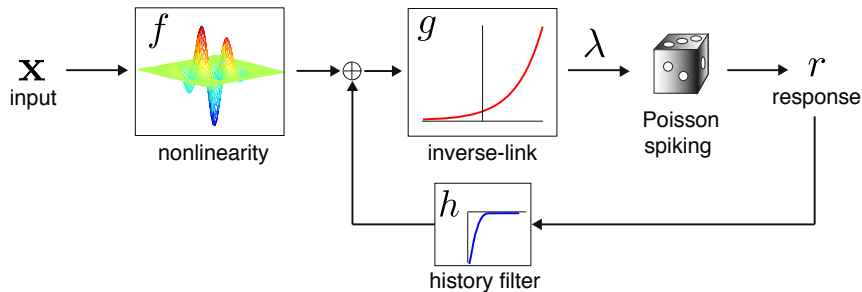

Figure 1: Encoding model schematic. The nonlinear function $f$ converts an input vector $\mathbf{x}$ to a scalar, which $g$ then transforms to a non-negative spike rate $\lambda = g(f(\mathbf{x}))$. The spike response $r$ is a Poisson random variable with mean $\lambda$.

dimensions (e.g., histograms, splines) [7, 12]; or nonlinearities defined by a sum or product of 1D nonlinear functions [10, 13].

In this paper, we use a Gaussian process (GP) to provide a flexible, computationally tractable model of the multi-dimensional neural response nonlinearity $f(\mathbf{x})$, where $\mathbf{x}$ is a vector in feature space. Intuitively, a GP defines a probability distribution over the infinite-dimensional space of functions by specifying a Gaussian distribution over its finite-dimensional marginals (i.e., the probability over the function values at any finite collection of points), with hyperparameters that control the function's variability and smoothness [14]. Although exact inference under a model with GP prior and Poisson observations is analytically intractable, a variety of approximate and sampling-based inference methods have been developed [15, 16]). Our work builds on a substantial literature in neuroscience that has used GP-based models to decode spike trains [17–19], estimate spatial receptive fields [20,21], infer continuous spike rates from spike trains [22–24], infer common inputs [25], and extract low-dimensional latent variables from multi-neuron spiking activity [26, 27].

We focus on data from trial-based experiments where stimulus-response pairs $(\mathbf{x}, r)$ are sparse in the space of possible stimuli. We use a fixed *inverse link function* $g$ to transform $f(\mathbf{x})$ to a non-negative spike rate, which ensures the posterior over $f$ is log-concave [6, 20]. This log-concavity justifies a Gaussian approximation to the posterior, which we use to perform rapid empirical Bayes estimation of hyperparameters [5, 28]. Our main contribution is an algorithm for optimal experimental design, which allows $f$ to be characterized quickly and accurately from limited data [29, 30]. The method relies on *uncertainty sampling* [31], which involves selecting the stimulus $\mathbf{x}$ for which $g(f(\mathbf{x}))$ is maximally uncertain given the data collected in the experiment so far. We apply our methods to the nonlinear color-tuning properties of macaque V1 neurons. We show that the GP-Poisson model provides a flexible, tractable model for these responses, and that optimal design can substantially reduce the number of stimuli required to characterize them.

## 2 GP-Poisson neural encoding model

### 2.1 Encoding model (likelihood)

We begin by defining a probabilistic encoding model for the neural response. Let $r_i$ be an observed neural response (the spike count in some time interval $T$) at the $i$'th trial given the input stimulus $\mathbf{x}_i$. Here, we will assume that $\mathbf{x}$ is $D$-dimensional vector in the moderately low-dimensional neural feature space to which the neuron is sensitive, the output of the "L" stage in the LNP model.

Under the encoding model (Fig. 1), an input vector $\mathbf{x}_i$ passes through a nonlinear function $f$, whose real-valued output is transformed to a positive spike rate through a (fixed) function $g$. The spike response is a Poisson random variable with mean $g(f(\mathbf{x}))$, so the conditional probability of a stimulus-response pair is Poisson:

$$p(r_i|\mathbf{x}_i, f) = \tfrac{1}{r_i!}\lambda_i^{r_i} e^{-\lambda_i}, \qquad \lambda_i = g(f(\mathbf{x}_i)). \tag{1}$$

For a complete dataset, the log-likelihood is:

$$\mathcal{L}(f) = \log p(\mathbf{r}|X, f) = \mathbf{r}^\top \log(g(\mathbf{f})) - \mathbf{1}^\top g(\mathbf{f}) + const, \tag{2}$$

where $\mathbf{r} = (r_1, \ldots, r_N)^\top$ is a vector of spike responses, $\mathbf{1}$ is a vector of ones, and $\mathbf{f} = (f(\mathbf{x}_1), \ldots f(\mathbf{x}_N))^\top$ is shorthand for the vector defined by evaluating $f$ at the points in $X = \{\mathbf{x}_1, \ldots \mathbf{x}_N\}$. Note that although $f$ is an infinite-dimensional object in the space of functions, the likelihood only depends on the value of $f$ at the points in $X$.

In this paper, we fix the inverse-link function to $g(f) = \log(1 + \exp(f))$, which has the nice property that it grows linearly for large $f$ and decays gracefully to zero for negative $f$. This allows us to place a Gaussian prior on $f$ without allocating probability mass to negative spike rates, and obviates the need for constrained optimization of $f$ (but see [22] for a highly efficient solution). Most importantly, for any $g$ that is simultaneously convex and log-concave[1], the log-likelihood $\mathcal{L}(f)$ is concave in $f$, meaning it is free of non-global local extrema [6,20]. Combining $\mathcal{L}$ with a log-concave prior (as we do in the next section) ensures the log-posterior is also concave.

## 2.2 Gaussian Process prior

Gaussian processes (GPs) allow us to define a probability distribution over the infinite-dimensional space of functions by specifying a Gaussian distribution over a function's finite-dimensional marginals (i.e., the probability over the function values at any finite collection of points). The hyperparameters defining this prior are a mean $\mu_f$ and a *kernel* function $k(\mathbf{x}_i, \mathbf{x}_j)$ that specifies the covariance between function values $f(\mathbf{x}_i)$ and $f(\mathbf{x}_j)$ for any pair of input points $\mathbf{x}_i$ and $\mathbf{x}_j$. Thus, the GP prior over the function values $\mathbf{f}$ is given by

$$p(\mathbf{f}) = \mathcal{N}(\mathbf{f} \,|\, \mu_f \mathbf{1}, K) = |2\pi K|^{-\frac{1}{2}} \exp\left(-\tfrac{1}{2}(\mathbf{f} - \mu_f \mathbf{1})^\top K^{-1}(\mathbf{f} - \mu_f \mathbf{1})\right) \quad (3)$$

where $K$ is a covariance matrix whose $i, j$'th entry is $K_{ij} = k(\mathbf{x}_i, \mathbf{x}_j)$. Generally, the kernel controls the prior smoothness of $f$ by determining how quickly the correlation between nearby function values falls off as a function of distance. (See [14] for a general treatment). Here, we use a Gaussian kernel, since neural response nonlinearities are expected to be smooth in general:

$$k(\mathbf{x}_i, \mathbf{x}_j) = \rho \exp\left(-||\mathbf{x}_i - \mathbf{x}_j||^2/(2\tau)\right), \quad (4)$$

where hyperparameters $\rho$ and $\tau$ control the marginal variance and smoothness scale, respectively. The GP therefore has three total hyperparameters, $\theta = \{\mu_f, \rho, \tau\}$ which set the prior mean and covariance matrix over $\mathbf{f}$ for any collection of points in $X$.

## 2.3 MAP inference for $f$

The *maximum a posteriori* (MAP) estimate can be obtained by numerically maximizing the posterior for $f$. From Bayes rule, the log-posterior is simply the sum of the log-likelihood (eq. 2) and log-prior (eq. 3) plus a constant:

$$\log p(\mathbf{f}|\mathbf{r}, X, \theta) = \mathbf{r}^\top \log(g(\mathbf{f})) - \mathbf{1}^\top g(\mathbf{f}) - \tfrac{1}{2}(\mathbf{f} - \mu_f)^\top K^{-1}(\mathbf{f} - \mu_f) + const. \quad (5)$$

As noted above, this posterior has a unique maximum $\mathbf{f_{map}}$ so long as $g$ is convex and log-concave.

However, the solution vector $\mathbf{f_{map}}$ defined this way contains only the function values at the points in the training set $X$. How do we find the MAP estimate of $f$ at other points not in our training set? The GP prior provides a simple analytic formula for the maximum of the joint marginal containing the training data and any new point $f^* = f(\mathbf{x}^*)$, for a new stimulus $\mathbf{x}^*$. We have

$$p(f^*, \mathbf{f}|\mathbf{x}^*, \mathbf{r}, X, \theta) = p(f^*|\mathbf{f}, \theta)p(\mathbf{f}|\mathbf{r}, X, \theta) = \mathcal{N}(f^*|\mu^*, \sigma^{*2})\, p(\mathbf{f}|\mathbf{r}, X, \theta) \quad (6)$$

where, from the GP prior, $\mu^* = \mu_f + \mathbf{k}^{*\top} K^{-1}(\mathbf{f} - \mu_f)$ and $\sigma^{*2} = k(\mathbf{x}^*, \mathbf{x}^*) - \mathbf{k}^{*\top} K^* \mathbf{k}^*$ are the ($\mathbf{f}$-dependent) mean and variance of $f^*$, and row vector $\mathbf{k}^* = (k(\mathbf{x}_1, \mathbf{x}^*), \ldots k(\mathbf{x}_N, \mathbf{x}^*))$. This factorization arises from the fact that $f^*$ is conditionally independent of the data given the value of the function at $X$. Clearly, this posterior marginal (eq. 6) is maximized when $f^* = \mu^*$ and $\mathbf{f} = \mathbf{f_{map}}$.[2] Thus, for any collection of novel points $X^*$, the MAP estimate for $f(X^*)$ is given by the mean of the conditional distribution over $\mathbf{f}^*$ given $\mathbf{f_{map}}$:

$$p(f(X^*)|X^*, \mathbf{f_{map}}, \theta) = \mathcal{N}\left(\mu_f + K^* K^{-1}(\mathbf{f_{map}} - \mu_f), \; K^{**} - K^* K^{-1} K^{*\top}\right) \quad (7)$$

where $K_{il}^* = k(\mathbf{x}_i^*, \mathbf{x}_l)$ and $K_{ij}^{**} = k(\mathbf{x}_i^*, \mathbf{x}_j^*)$.

In practice, the prior covariance matrix $K$ is often ill-conditioned when datapoints in $X$ are closely spaced and smoothing hyperparameter $\tau$ is large, making it impossible to numerically compute $K^{-1}$. When the number of points is not too large ($N < 1000$), we can address this by performing a singular value decomposition (SVD) of $K$ and keeping only the singular vectors with singular value above some threshold. This results in a lower-dimensional numerical optimization problem, since we only have to search the space spanned by the singular vectors of $K$. We discuss strategies for scaling to larger datasets in the Discussion.

## 2.4 Efficient evidence optimization for $\theta$

The hyperparameters $\theta = \{\mu_f, \rho, \tau\}$ that control the GP prior have a major influence on the shape of the inferred nonlinearity, particularly in high dimensions and when data is scarce. A theoretically attractive and computationally efficient approach for setting $\theta$ is to maximize the *evidence* $p(\theta|\mathbf{r}, X)$, also known as the *marginal likelihood*, a general approach known as *empirical Bayes* [5, 14, 28, 32]. Here we describe a method for rapid evidence maximization that we will exploit to design an active learning algorithm in Section 3.

The evidence can be computed by integrating the product of the likelihood and prior with respect to $f$, but can also be obtained by solving for the (often neglected) denominator term in Bayes' rule:

$$p(\mathbf{r}|\theta) = \int p(\mathbf{r}|\mathbf{f})p(f|\theta)d\mathbf{f} = \frac{p(\mathbf{r}|\mathbf{f})p(\mathbf{f}|\theta)}{p(\mathbf{f}|\mathbf{r}, \theta)}, \qquad (8)$$

where we have dropped conditioning on $X$ for notational convenience. For the GP-Poisson model here, this integral is not tractable analytically, but we can approximate it as follows. We begin with a well-known Gaussian approximation to the posterior known as the *Laplace approximation*, which comes from a 2nd-order Taylor expansion of the log-posterior around its maximum [28]:

$$p(\mathbf{f}|\mathbf{r}, \theta) \approx \mathcal{N}(\mathbf{f}|\mathbf{f_{map}}, \Lambda), \qquad \Lambda^{-1} = H + K^{-1}, \qquad (9)$$

where $H = \frac{\partial^2}{\partial \mathbf{f}^2}\mathcal{L}(\mathbf{f})$ is the Hessian (second derivative matrix) of the negative log-likelihood (eq. 2), evaluated at $\mathbf{f_{map}}$, and $K^{-1}$ is the inverse prior covariance (eq. 3). This approximation is reasonable given that the posterior is guaranteed to be unimodal and log-concave. Plugging it into the denominator in (eq. 8) gives us a formula for evaluating approximate evidence,

$$p(\mathbf{r}|\theta) \approx \frac{\exp\left(\mathcal{L}(\mathbf{f})\right)\mathcal{N}(\mathbf{f}|\boldsymbol{\mu_f}, K)}{\mathcal{N}(\mathbf{f}|\mathbf{f_{map}}, \Lambda)}, \qquad (10)$$

which we evaluate at $\mathbf{f} = \mathbf{f_{map}}$, since the Laplace approximation is the most accurate there [20, 33].

The hyperparameters $\theta$ directly affect the prior mean and covariance ($\boldsymbol{\mu_f}, K$), as well as the posterior mean and covariance ($\mathbf{f_{map}}, \Lambda$), all of which are essential for evaluating the evidence. Finding $\mathbf{f_{map}}$ and $\Lambda$ given $\theta$ requires numerical optimization of $\log p(\mathbf{f}|r, \theta)$, which is computationally expensive to perform for each search step in $\theta$. To overcome this difficulty, we decompose the posterior moments ($\mathbf{f_{map}}, \Lambda$) into terms that depend on $\theta$ and terms that do not via a Gaussian approximation to the likelihood. The logic here is that a Gaussian posterior and prior imply a likelihood function proportional to a Gaussian, which in turn allows prior and posterior moments to be computed analytically for each $\theta$. This trick is similar to that of the EP algorithm [34]: we divide a Gaussian component out of the Gaussian posterior and approximate the remainder as Gaussian. The resulting moments are $H = \Lambda^{-1} - K^{-1}$ for the likelihood inverse-covariance (which is the Hessian of the log-likelihood from eq. 9), and $\mathbf{m} = H^{-1}(\Lambda^{-1}\mathbf{f_{map}} - K^{-1}\boldsymbol{\mu_f})$ for the likelihood mean, which comes from the standard formula for the product of two Gaussians.

Our algorithm for evidence optimization proceeds as follows: **(1)** given the current hyperparameters $\theta_i$, numerically maximize the posterior and form the Laplace approximation $\mathcal{N}(\mathbf{f_{map}}_i, \Lambda_i)$; **(2)** compute the Gaussian "potential" $\mathcal{N}(\mathbf{m}_i, H_i)$ underlying the likelihood, given the current values of ($\mathbf{f_{map}}_i, \Lambda_i, \theta_i$), as described above; **(3)** Find $\theta_{i+1}$ by maximizing the log-evidence, which is:

$$\mathcal{E}(\theta) = \mathbf{r}^T \log(g(\mathbf{f_{map}})) - \mathbf{1}^T g(\mathbf{f_{map}}) - \frac{1}{2}\log|KH_i + I| - \frac{1}{2}(\mathbf{f_{map}} - \boldsymbol{\mu_f})^T K^{-1}(\mathbf{f_{map}} - \boldsymbol{\mu_f}), \quad (11)$$

where $\mathbf{f_{map}}$ and $\Lambda$ are updated using $H_i$ and $\mathbf{m}_i$ obtained in step **(2)**, i.e. $\mathbf{f_{map}} = \Lambda(H_i\mathbf{m}_i + K^{-1}\boldsymbol{\mu_f})$ and $\Lambda = (H_i + K^{-1})^{-1}$. Note that this significantly expedites evidence optimization since we do not have to numerically optimize $\mathbf{f_{map}}$ for each $\theta$.

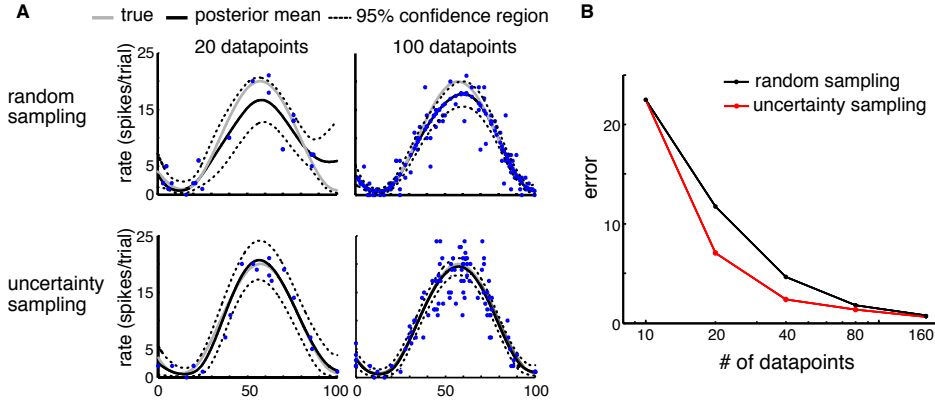

Figure 2: Comparison of random and optimal design in a simulated experiment with a 1D nonlinearity. The true nonlinear response function $g(f(x))$ is in gray, the posterior mean is in black solid, 95% confidence interval is in black dotted, stimulus is in blue dots. **A (top)**: Random design: responses were measured with 20 (left) and 100 (right) additional stimuli, with stimuli sampled uniformly over the interval shown on the $x$ axis. **A (bottom)**: Optimal design: responses were measured with same numbers of additional stimuli selected by uncertainty sampling (see text). **B**: Mean square error as a function of the number of stimulus-response pairs. The optimal design achieved half the error rate of the random design experiment.

# 3  Optimal design: uncertainty sampling

So far, we have introduced an efficient algorithm for estimating the nonlinearity $f$ and hyperparameters $\theta$ for an LNP encoding model under a GP prior. Here we introduce a method for adaptively selecting stimuli during an experiment (often referred to as *active learning* or *optimal experimental design*) to minimize the amount of data required to estimate $f$ [29]. The basic idea is that we should select stimuli that maximize the expected information gained about the model parameters. This information gain of course depends the posterior distribution over the parameters given the data collected so far. *Uncertainty sampling* [31] is an algorithm that is appropriate when the model parameters and stimulus space are in a 1-1 correspondence. It involves selecting the stimulus $\mathbf{x}$ for which the posterior over parameter $f(\mathbf{x})$ has highest entropy, which in the case of a Gaussian posterior corresponds to the highest posterior variance.

Here we alter the algorithm slightly to select stimuli for which we are most uncertain about the spike rate $g(f(\mathbf{x}))$, *not* (as stated above) the stimuli where we are most uncertain about our underlying function $f(\mathbf{x})$. The rationale for this approach is that we are generally more interested in the neuron's spike-rate as a function of the stimulus (which involves the inverse link function $g$) than in the parameters we have used to define that function. Moreover, any link function that maps $\mathbb{R}$ to the positive reals $\mathbb{R}^+$, as required for Poisson models, we will have unavoidable uncertainty about negative values of $f$, which will not be overcome by sampling small (integer) spike-count responses. Our strategy therefore focuses on uncertainty in the expected spike-rate rather than uncertainty in $f$.

Our method proceeds as follows. Given the data observed up to a certain time in the experiment, we define a grid of (evenly-spaced) points $\{\mathbf{x}_j^*\}$ as candidate next stimuli. For each point, we compute the posterior uncertainty $\gamma_j$ about the spike rate $g(f(\mathbf{x}_j^*))$ using the delta method, i.e., $\gamma_j = g'(f(\mathbf{x}_j^*))\sigma_j$, where $\sigma_j$ is the posterior standard deviaton (square root of the posterior variance) at $f(\mathbf{x}_j)$ and $g'$ is the derivative of $g$ with respect to its argument. The stimulus selected next on trial $t+1$, given all data observed up to time $t$, is selected randomly from the set:

$$\mathbf{x}_{t+1} \in \{\mathbf{x}_j^* \mid \gamma_j \geq \gamma_i \forall i\}, \tag{12}$$

that is, the set of all stimuli for which uncertainty $\gamma$ is maximal. To find $\{\sigma_j\}$ at each candidate point, we must first update $\theta$ and $\mathbf{f_{map}}$. After each trial, we update $\mathbf{f_{map}}$ by numerically optimizing the posterior, then update the hyperparameters using (eq. 11), and then numerically re-compute $\mathbf{f_{map}}$ and $\Lambda$ given the new $\theta$. The method is summarized in Algorithm 1, and runtimes are shown in Fig. 5.

**Algorithm 1** Optimal design for nonlinearity estimation under a GP-Poisson model

1. given the current data $D_t = \{\mathbf{x}_1, ..., \mathbf{x}_t, r_1, ..., r_t\}$, the posterior mode $\mathbf{f}_{\mathbf{map}_t}$, and hyper-parameters $\theta_t$, compute the posterior mean and standard deviation $(\mathbf{f}_{\mathbf{map}}^*, \boldsymbol{\sigma}^*)$ at a grid of candidate stimulus locations $\{\mathbf{x}^*\}$.

2. select the element of $\{\mathbf{x}^*\}$ for which $\gamma^* = g'(\mathbf{f}_{\mathbf{map}}^*)\boldsymbol{\sigma}^*$ is maximal

3. present the selected $\mathbf{x}_{t+1}$ and record the neural response $r_{t+1}$

4. find $\mathbf{f}_{\mathbf{map}_{t+1}} | D_{t+1}, \theta_t$; update $\theta_{i+1}$ by maximizing evidence; find $\mathbf{f}_{\mathbf{map}_{t+1}} | D_{t+1}, \theta_{t+1}$

## 4 Simulations

We tested our method in simulation using a 1-dimensional feature space, where it is easy to visualize the nonlinearity and the uncertainty of our estimates (Fig. 2). The stimulus space was taken to be the range $[0, 100]$, the true $f$ was a sinusoid, and spike responses were simulated as Poisson with rate $g(f(x))$. We compared the estimate of $g(f(x))$ obtained using optimal design to the estimate obtained with "random sampling", stimuli drawn uniformly from the stimulus range.

Fig. 2 shows the estimates of $g(f(x))$ after 20 and 100 trials using each method, along with the marginal posterior standard deviation, which provides a $\pm 2$ SD Bayesian confidence interval for the estimate. The optimal design method effectively decreased the high variance in the middle (near 50) because it drew more samples where uncertainty about the spike rate was higher (due to the fact that variance increases with mean for Poisson neurons). The estimates using random sampling (A, top) was not accurate because it drew more points in the tails where the variance was originally lower than the center. We also examined the errors in each method as a function of the number of data points. We drew each number of data points 100 times and computed the average error between the estimate and the true $g(f(x))$. As shown in (B), uncertainty sampling achieved roughly half the error rate of the random sampling after 20 datapoints.

## 5 Experiments

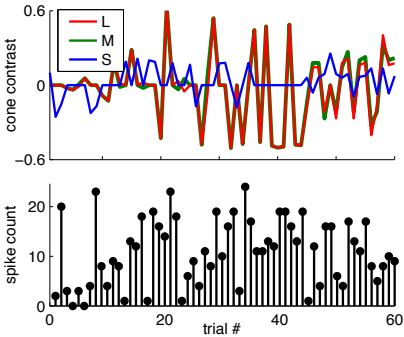

Figure 3: Raw experimental data: stimuli in 3D cone-contrast space (above) and recorded spike counts (below) during the first 60 experimental trials. Several (3-6) stimulus staircases along different directions in color space were randomly interleaved to avoid the effects of adaptation; a color direction is defined as the relative proportions of L, M, and S cone contrasts, with [0 0 0] corresponding to a neutral gray (zero-contrast) stimulus. In each color direction, contrast was actively titrated with the aim of evoking a response of 29 spikes/sec. This sampling procedure permitted a broad survey of the stimulus space, with the objective that many stimuli evoked a statistically reliable but non-saturating response. In all, 677 stimuli in 65 color directions were presented for this neuron.

We recorded from a V1 neuron in an awake, fixating rhesus monkey while Gabor patterns with varying color and contrast were presented at the receptive field. Orientation and spatial frequency of the Gabor were fixed at preferred values for the neuron and drifted at 3 Hz for 667 ms each. Contrast was varied using multiple interleaved staircases along different axes in color space, and spikes were counted during a 557ms window beginning 100ms after stimulus appeared. The staircase design was used because the experiments were carried out prior to formulating the optimal design methods described in this paper. However, we will analyze them here for a "simulated optimal design experiment", where we choose stimuli sequentially from the list of stimuli that were actually presented during the experiment, in an order determined by our information-theoretic criterion. See Fig. 3 caption for more details of the experimental recording.

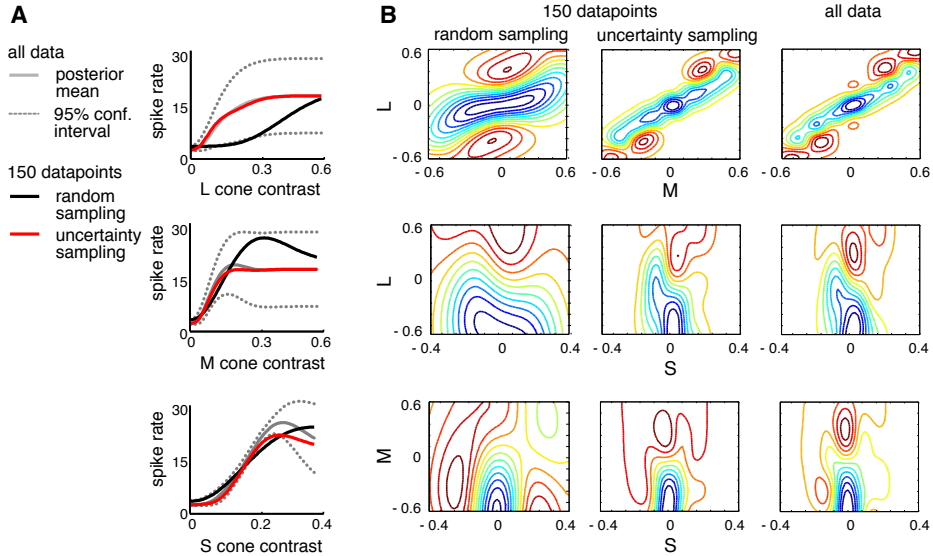

Figure 4: One and two-dimensional conditional "slices" through the 3D nonlinearity of a V1 simple cell in cone contrast space. **A**: 1D conditionals showing spike rate as a function of L, M, and S cone contrast, respectively, with other cone contrasts fixed to zero. Traces show the posterior mean and $\pm 2SD$ credible interval given all datapoints (solid and dotted gray), and the posterior mean given only 150 data points selected randomly (black) or by optimal design (red), carried out by drawing a subset of the data points actually collected during the experiment. Note that even with only 1/4 of data, the optimal design estimate is nearly identical to the estimate obtained from all 677 datapoints. **B**: 2D conditionals on M and L (first row), S and L (second row), M and S (third row) cones, respectively, with the other cone contrast set to zero. 2D conditionals using optimal design sampling (middle column) with 150 data points are much closer to the 2D conditionals using all data (right column) than those from a random sub-sampling of 150 points (left column).

We first used the entire dataset (677 stimulus-response pairs) to find the posterior maximum $\mathbf{f_{map}}$, with hyperparameters set by maximizing evidence (sequential optimization of $\mathbf{f_{map}}$ and $\theta$ (eq. 11) until convergence). Fig. 4 shows 1D and 2D conditional slices through the estimated 3D nonlinearity $g(f(\mathbf{x}))$, with contour plots constructed using the MAP estimate of $f$ on a fine grid of points. The contours for a neuron with linear summation of cone contrasts followed by an output nonlinearity (i.e., as assumed by the standard model of V1 simple cells) would consist of straight lines. The curvature observed in contour plots (Fig. 4B) indicates that cone contrasts are summed together in a highly nonlinear fashion, especially for L and M cones (top).

We then performed a simulated optimal design experiment by selecting from the 677 stimulus-response pairs collected during the experiment, and re-ordering them greedily according to the uncertainty sampling algorithm described above. We compared the estimate obtained using only 1/4 of the data (150 points) with an estimate obtained if we had randomly sub-sampled 150 data points from the dataset (Fig. 4). Using only 150 data points, the conditionals of the estimate using uncertainty sampling were almost identical to those using all data (677 points).

Although our software implementation of the optimal design method was crude (using Matlab's `fminunc` twice to find $\mathbf{f_{map}}$ and `fmincon` once to optimize the hyperparameters during each inter-trial interval), the speed was more than adequate for the experimental data collected (Fig. 5, A) using a machine with an Intel 3.33GHz XEON processor. The largest bottleneck by far was computing the eigendecomposition of $K$ for each search step for $\theta$. We will discuss briefly how to improve the speed of our algorithm in the Discussion.

Lastly, we added a recursive filter $h$ to the model (Fig. 1), to incorporate the effects of spike history on the neuron's response, allowing us to account for the possible effects of adaptation on the spike counts obtained. We computed the maximum a posteriori (MAP) estimate for $h$ under a temporal

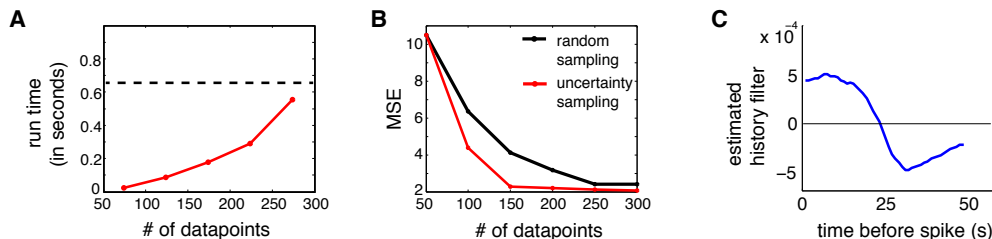

Figure 5: Comparison of run time and error of optimal design method using simulated experiments by resampling experimental data. **A**: The run time for uncertainty sampling (including the posterior update and the evidence optimization) as a function of the number of data points observed. (The grid of "candidate" stimuli $\{\mathbf{x}^*\}$ was the subset of stimuli in the experimental dataset not yet selected, but the speed was not noticeably affected by scaling to much larger sets of candidate stimuli). The black dotted line shows the mean intertrial interval of 677ms. **B**: The mean squared error between the estimate obtained using each sampling method and that obtained using the full dataset. Note that the error of uncertainty sampling with 150 points is even lower than that from random sampling with 300 data points. **C**: Estimated response-history filter $h$, which describes how recent spiking influences the neuron's spike rate. This neuron shows self-excitatory influence on the time-scale of 25s, with self-suppression on a longer scale of approximately 1m.

smoothing prior (Fig. 5). It shows that the neuron's response has a mild dependence on its recent spike-history, with a self-exciting effect of spikes within the last 25s. We evaluated the performance of the augmented model by holding out a random 10% of the data for cross-validation. Prediction performance on test data was more accurate by an average of 0.2 spikes per trial in predicted spike count, a 4 percent reduction in cross-validation error compared to the original model.

# 6   Discussion

We have developed an algorithm for optimal experimental design, which allows the nonlinearity in a cascade neural encoding model to be characterized quickly and accurately from limited data. The method relies on a fast method for updating the hyperparameters using a Gaussian factorization of the Laplace approximation to the posterior, which removes the need to numerically recompute the MAP estimate as we optimize the hyperparameters. We described a method for optimal experimental design, based on uncertainty sampling, to reduce the number of stimuli required to estimate such response functions. We applied our method to the nonlinear color-tuning properties of macaque V1 neurons and showed that the GP-Poisson model provides a flexible, tractable model for these responses, and that optimal design can substantially reduce the number of stimuli required to characterize them. One additional virtue of the GP-Poisson model is that conditionals and marginals of the high-dimensional nonlinearity are straightforward, making it easy to visualize their lower-dimensional slices and projections (as we have done in Fig. 4). We added a history term to the LNP model in order to incorporate the effects of recent spike history on the spike rate (Fig. 5), which provided a very slight improvement in prediction accuracy. We expect that the ability to incorporate dependencies on spike history to be important for the success of optimal design experiments, especially with neurons that exhibit strong spike-rate adaptation [30].

One potential criticism of our approach is that uncertainty sampling in unbounded spaces is known to "run away from the data", repeatedly selecting stimuli that are far from previous measurements. We wish to point out that in neural applications, the stimulus space is always bounded (e.g., by the gamut of the monitor), and in our case, stimuli at the corners of the space are actually helpful for initializing estimates the range and smoothness of the function.

In future work, we will work to improve the speed of the algorithm for use in real-time neurophysiology experiments, using analytic first and second derivatives for evidence optimization and exploring approximate methods for sparse GP inference [35]. We will examine kernel functions with a more tractable matrix inverse [20], and test other information-theoretic data selection criteria for response function estimation [36].

## Footnotes

[1] Such functions must grow monotonically at least linearly and at most exponentially [6]. Examples include exponential, half-rectified linear, $\log(1 + \exp(f))^p$ for $p \geq 1$.

[2] Note that this is not necessarily identical to the *marginal* MAP estimate of $f^*|\mathbf{x}^*, \mathbf{r}, X, \theta$, which requires maximizing (eq. 6) integrated with respect to $f$.

# References

[1] E. P. Simoncelli, J. W. Pillow, L. Paninski, and O. Schwartz. *The Cognitive Neurosciences, III*, chapter 23, pages 327–338. MIT Press, Cambridge, MA, October 2004.

[2] R.R. de Ruyter van Steveninck and W. Bialek. *Proc. R. Soc. Lond. B*, 234:379–414, 1988.

[3] E. J. Chichilnisky. *Network: Computation in Neural Systems*, 12:199–213, 2001.

[4] F. Theunissen, S. David, N. Singh, A. Hsu, W. Vinje, and J. Gallant. *Network: Computation in Neural Systems*, 12:289–316, 2001.

[5] M. Sahani and J. Linden. *NIPS*, 15, 2003.

[6] L. Paninski. *Network: Computation in Neural Systems*, 15:243–262, 2004.

[7] Tatyana Sharpee, Nicole C Rust, and William Bialek. *Neural Comput*, 16(2):223–250, Feb 2004.

[8] O. Schwartz, J. W. Pillow, N. C. Rust, and E. P. Simoncelli. *Journal of Vision*, 6(4):484–507, 7 2006.

[9] J. W. Pillow and E. P. Simoncelli. *Journal of Vision*, 6(4):414–428, 4 2006.

[10] Misha B Ahrens, Jennifer F Linden, and Maneesh Sahani. *J Neurosci*, 28(8):1929–1942, Feb 2008.

[11] Nicole C Rust, Odelia Schwartz, J. Anthony Movshon, and Eero P Simoncelli. *Neuron*, 46(6):945–956, Jun 2005.

[12] I. DiMatteo, C. Genovese, and R. Kass. *Biometrika*, 88:1055–1073, 2001.

[13] S.F. Martins, L.A. Sousa, and J.C. Martins. *Image Processing, 2007. ICIP 2007. IEEE International Conference on*, volume 3, pages III–309. IEEE, 2007.

[14] Carl Rasmussen and Chris Williams. *Gaussian Processes for Machine Learning*. MIT Press, 2006.

[15] Liam Paninski, Yashar Ahmadian, Daniel Gil Ferreira, Shinsuke Koyama, Kamiar Rahnama Rad, Michael Vidne, Joshua Vogelstein, and Wei Wu. *J Comput Neurosci*, Aug 2009.

[16] Jarno Vanhatalo, Ville Pietiläinen, and Aki Vehtari. *Statistics in medicine*, 29(15):1580–1607, July 2010.

[17] E. Brown, L. Frank, D. Tang, M. Quirk, and M. Wilson. *Journal of Neuroscience*, 18:7411–7425, 1998.

[18] W. Wu, Y. Gao, E. Bienenstock, J.P. Donoghue, and M.J. Black. *Neural Computation*, 18(1):80–118, 2006.

[19] Y. Ahmadian, J. W. Pillow, and L. Paninski. *Neural Comput*, 23(1):46–96, Jan 2011.

[20] K.R. Rad and L. Paninski. *Network: Computation in Neural Systems*, 21(3-4):142–168, 2010.

[21] Jakob H Macke, Sebastian Gerwinn, Leonard E White, Matthias Kaschube, and Matthias Bethge. *Neuroimage*, 56(2):570–581, May 2011.

[22] John P. Cunningham, Krishna V. Shenoy, and Maneesh Sahani. *Proceedings of the 25th international conference on Machine learning*, ICML '08, pages 192–199, New York, NY, USA, 2008. ACM.

[23] R.P. Adams, I. Murray, and D.J.C. MacKay. *Proceedings of the 26th Annual International Conference on Machine Learning*. ACM New York, NY, USA, 2009.

[24] Todd P. Coleman and Sridevi S. Sarma. *Neural Computation*, 22(8):2002–2030, 2010.

[25] J. E. Kulkarni and L Paninski. *Network: Computation in Neural Systems*, 18(4):375–407, 2007.

[26] A.C. Smith and E.N. Brown. *Neural Computation*, 15(5):965–991, 2003.

[27] B.M. Yu, J.P. Cunningham, G. Santhanam, S.I. Ryu, K.V. Shenoy, and M. Sahani. *Journal of Neurophysiology*, 102(1):614, 2009.

[28] C.M. Bishop. *Pattern recognition and machine learning*. Springer New York:, 2006.

[29] D. Mackay. *Neural Computation*, 4:589–603, 1992.

[30] J. Lewi, R. Butera, and L. Paninski. *Neural Computation*, 21(3):619–687, 2009.

[31] David D. Lewis and William A. Gale. *Proceedings of the ACM SIGIR conference on Research and Development in Information Retrieval*, pages 3–12. Springer-Verlag, 1994.

[32] G. Casella. *American Statistician*, pages 83–87, 1985.

[33] J. W. Pillow, Y. Ahmadian, and L. Paninski. *Neural Comput*, 23(1):1–45, Jan 2011.

[34] T. P. Minka. *UAI '01: Proceedings of the 17th Conference in Uncertainty in Artificial Intelligence*, pages 362–369, San Francisco, CA, USA, 2001. Morgan Kaufmann Publishers Inc.

[35] E. Snelson and Z. Ghahramani. *Advances in neural information processing systems*, 18:1257, 2006.

[36] Andreas Krause, Ajit Singh, and Carlos Guestrin. *J. Mach. Learn. Res.*, 9:235–284, June 2008.

